# Ranking Measures and Loss Functions in Learning to Rank

**Wei Chen**[*]
Chinese Academy of sciences
chenwei@amss.ac.cn

**Tie-Yan Liu**
Microsoft Research Asia
tyliu@micorsoft.com

**Yanyan Lan**
Chinese Academy of sciences
lanyanyan@amss.ac.cn

**Zhiming Ma**
Chinese Academy of sciences
mazm@amt.ac.cn

**Hang Li**
Microsoft Research Asia
hangli@micorsoft.com

## Abstract

Learning to rank has become an important research topic in machine learning. While most learning-to-rank methods learn the ranking functions by minimizing loss functions, it is the ranking measures (such as NDCG and MAP) that are used to evaluate the performance of the learned ranking functions. In this work, we reveal the relationship between ranking measures and loss functions in learning-to-rank methods, such as Ranking SVM, RankBoost, RankNet, and ListMLE. We show that the loss functions of these methods are upper bounds of the measure-based ranking errors. As a result, the minimization of these loss functions will lead to the maximization of the ranking measures. The key to obtaining this result is to model ranking as a sequence of classification tasks, and define a so-called *essential loss* for ranking as the weighted sum of the classification errors of individual tasks in the sequence. We have proved that the essential loss is both an upper bound of the measure-based ranking errors, and a lower bound of the loss functions in the aforementioned methods. Our proof technique also suggests a way to modify existing loss functions to make them tighter bounds of the measure-based ranking errors. Experimental results on benchmark datasets show that the modifications can lead to better ranking performances, demonstrating the correctness of our theoretical analysis.

## 1 Introduction

Learning to rank has become an important research topic in many fields, such as machine learning and information retrieval. The process of learning to rank is as follows. In training, a number of sets are given, each set consisting of objects and labels representing their rankings (e.g., in terms of multi-level ratings[1]). Then a ranking function is constructed by minimizing a certain loss function on the training data. In testing, given a new set of objects, the ranking function is applied to produce a ranked list of the objects.

Many learning-to-rank methods have been proposed in the literature, with different motivations and formulations. In general, these methods can be divided into three categories [3]. The *pointwise* approach, such as subset regression [5] and McRank [10], views each single object as the learning instance. The *pairwise* approach, such as Ranking SVM [7], RankBoost [6], and RankNet [2], regards a pair of objects as the learning instance. The *listwise* approach, such as ListNet [3] and

---

[*]The work was performed when the first and the third authors were interns at Microsoft Research Asia.
[1]In information retrieval, such a label represents the relevance of a document to the given query.

ListMLE [16], takes the entire ranked list of objects as the learning instance. Almost all these methods learn their ranking functions by minimizing certain loss functions, namely the pointwise, pairwise, and listwise losses. On the other hand, however, it is the ranking measures that are used to evaluate the performance of the learned ranking functions. Taking information retrieval as an example, measures such as Normalized Discounted Cumulative Gain (NDCG) [8] and Mean Average Precision (MAP) [1] are widely used, which obviously differ from the loss functions used in the aforementioned methods. In such a situation, a natural question to ask is *whether the minimization of the loss functions can really lead to the optimization of the ranking measures*.[2]

Actually people have tried to answer this question. It has been proved in [5] and [10] that the regression and classification based losses used in the pointwise approach are upper bounds of $(1-\text{NDCG})$. However, for the pairwise and listwise approaches, which are regarded as the state-of-the-art of learning to rank [3, 11], limited results have been obtained. The motivation of this work is to reveal the relationship between ranking measures and the pairwise/listwise losses.

The problem is non-trivial to solve, however. Note that ranking measures like NDCG and MAP are defined with the labels of objects (i.e., in terms of multi-level ratings). Therefore it is relatively easy to establish the connection between the pointwise losses and the ranking measures, since the pointwise losses are also defined with the labels of objects. In contrast, the pairwise and listwise losses are defined with the partial or total order relations among objects, rather than their individual labels. As a result, it is much more difficult to bridge the gap between the pairwise/listwise losses and the ranking measures.

To tackle the challenge, we propose making a transformation of the labels on objects to a permutation set. All the permutations in the set are consistent with the labels, in the sense that an object with a higher rating is ranked before another object with a lower rating in the permutation. We then define an *essential loss* for ranking on the permutation set as follows. First, for each permutation, we construct a sequence of classification tasks, with the goal of each task being to distinguish an object from the objects ranked below it in the permutation. Second, the weighted sum of the classification errors of individual tasks in the sequence is computed. Third, the essential loss is defined as the minimum value of the weighted sum over all the permutations in the set.

Our study shows that the essential loss has several nice properties, which help us reveal the relationship between ranking measures and the pairwise/listwise losses. First, it can be proved that the essential loss is an upper bound of measure-based ranking errors such as $(1-\text{NDCG})$ and $(1-\text{MAP})$. Furthermore, the zero value of the essential loss is a *sufficient* and *necessary* condition for the zero values of $(1-\text{NDCG})$ and $(1-\text{MAP})$. Second, it can be proved that the pairwise losses in Ranking SVM, RankBoost, and RankNet, and the listwise loss in ListMLE are all upper bounds of the essential loss. As a consequence, we come to the conclusion that the loss functions used in these methods can bound $(1-\text{NDCG})$ and $(1-\text{MAP})$ from above. In other words, the minimization of these loss functions can effectively maximize NDCG and MAP.

The proofs of the above results suggest a way to modify existing pairwise/listwise losses so as to make them tighter bounds of $(1-\text{NDCG})$. We hypothesize that tighter bounds will lead to better ranking performances; we tested this hypothesis using benchmark datasets. The experimental results show that the methods minimizing the modified losses can outperform the original methods, as well as many other baseline methods. This validates the correctness of our theoretical analysis.

## 2  Related work

In this section, we review the widely-used loss functions in learning to rank, ranking measures in information retrieval, and previous work on the relationship between loss functions and ranking measures.

## 2.1 Loss functions in learning to rank

Let $\mathbf{x} = \{x_1, \cdots, x_n\}$ be the objects be to ranked.[3] Suppose the labels of the objects are given as multi-level ratings $\mathcal{L} = \{l(1), ..., l(n)\}$, where $l(i) \in \{r_1, ..., r_K\}$ denotes the label of $x_i$ [11]. Without loss of generality, we assume $l(i) \in \{0, 1, ..., K - 1\}$ and name the corresponding labels as $K$-level ratings. If $l(i) > l(j)$, then $x_i$ should be ranked before $x_j$. Let $\mathcal{F}$ be the function class and $f \in \mathcal{F}$ be a ranking function. The optimal ranking function is learned from the training data by minimizing a certain loss function defined on the objects, their labels, and the ranking function. Several approaches have been proposed to learn the optimal ranking function.

In the *pointwise approach*, the loss function is defined on the basis of single objects. For example, in subset regression [5], the loss function is as follows,

$$L^r(f; \mathbf{x}, \mathcal{L}) = \sum_{i=1}^{n} \big( f(x_i) - l(i) \big)^2. \tag{1}$$

In the *pairwise approach*, the loss function is defined on the basis of pairs of objects whose labels are different. For example, the loss functions of Ranking SVM [7], RankBoost [6], and RankNet [2] all have the following form,

$$L^p(f; \mathbf{x}, \mathcal{L}) = \sum_{s=1}^{n-1} \sum_{i=1, l(i) < l(s)}^{n} \phi\big(f(x_s) - f(x_i)\big), \tag{2}$$

where the $\phi$ functions are hinge function ($\phi(z) = (1 - z)_+$), exponential function ($\phi(z) = e^{-z}$), and logistic function ($\phi(z) = \log(1 + e^{-z})$) respectively, for the three algorithms.

In the *listwise approach*, the loss function is defined on the basis of all the $n$ objects. For example, in ListMLE [16], the following loss function is used,

$$L^l(f; \mathbf{x}, y) = \sum_{s=1}^{n-1} \Big( - f(x_{y(s)}) + \ln \big( \sum_{i=s}^{n} \exp(f(x_{y(i)})) \big) \Big), \tag{3}$$

where $y$ is a randomly selected permutation (i.e., ranked list) that satisfies the following condition: for any two objects $x_i$ and $x_j$, if $l(i) > l(j)$, then $x_i$ is ranked before $x_j$ in $y$. Notation $y(i)$ represents the index of the object ranked at the $i$-th position in $y$.

## 2.2 Ranking measures

Several ranking measures have been proposed in the literature to evaluate the performance of a ranking function. Here we introduce two of them, NDCG [8] and MAP[1], which are popularly used in information retrieval.

NDCG is defined with respect to $K$-level ratings $\mathcal{L}$,

$$NDCG(f; \mathbf{x}, \mathcal{L}) = \frac{1}{N_n} \sum_{r=1}^{n} G\big(l(\pi_f(r))\big) D(r),$$

where $\pi_f$ is the ranked list produced by ranking function $f$, $G$ is an increasing function (named the gain function), $D$ is a decreasing function (named the position discount function), and $N_n = \max_\pi \sum_{r=1}^{n} G\big(l(\pi(r))\big) D(r)$. In practice, one usually sets $G(z) = 2^z - 1$; $D(z) = \frac{1}{\log_2(1+z)}$ if $z \leq C$, and $D(z) = 0$ if $z > C$ ($C$ is a fixed integer).

MAP is defined with respect to 2-level ratings as follows,

$$MAP(f; \mathbf{x}, \mathcal{L}) = \frac{1}{n_1} \sum_{s:l(\pi_f(s))=1} \frac{\sum_{i \leq s} I_{\{l(\pi_f(i))=1\}}}{s}. \tag{4}$$

where $I_{\{\cdot\}}$ is the indicator function, and $n_1$ is the number of objects with label 1. When the labels are given in terms of $K$-level ratings ($K > 2$), a common practice of using MAP is to fix a level $k^*$, and regard all the objects whose levels are lower than $k^*$ as having label 0, and regard the other objects as having label 1 [11].

From the definitions of NDCG and MAP, we can see that their maximum values are both one. Therefore, we can consider $(1 - \text{NDCG})$ and $(1 - \text{MAP})$ as ranking errors. For ease of reference, we call them *measure-based ranking errors*.

### 2.3 Previous bounds

For the pointwise approach, the following results have been obtained in [5] and [10].[4]

The regression based pointwise loss is an upper bound of $(1-NDCG)$,

$$1 - NDCG(f; \mathbf{x}, \mathcal{L}) \leq \frac{1}{N_n} \Big( 2 \sum_{i=1}^{n} D(i)^2 \Big)^{1/2} L^r(f; \mathbf{x}, \mathcal{L})^{1/2}.$$

The classification based pointwise loss is also an upper bound of $(1-NDCG)$,

$$1 - NDCG(f; \mathbf{x}, \mathcal{L}) \leq \frac{15\sqrt{2}}{N_n} \Big( \sum_{i=1}^{n} D(i)^2 - n \prod_{i=1}^{n} D(i)^{2/n} \Big)^{1/2} \Big( \sum_{i=1}^{n} I_{\{\hat{l}(i) \neq l(i)\}} \Big)^{1/2},$$

where $\hat{l}(i)$ is the label of object $x_i$ predicted by the classifier, in the setting of 5-level ratings.
For the pairwise approach, the following result has been obtained [9],

$$1 - MAP(f; \mathbf{x}, \mathcal{L}) \leq 1 - \frac{1}{n_1} (L^p(f; \mathbf{x}, \mathcal{L}) + C_{n_1+1}^2)^{-1} (\sum_{i=1}^{n_1} \sqrt{i})^2.$$

According to the above results, minimizing the regression and classification based pointwise losses will minimize $(1-NDCG)$. Note that the zero values of these two losses are sufficient but not necessary conditions for the zero value of $(1-NDCG)$. That is, when $(1-NDCG)$ is zero, the loss functions may still be very large [10]. For the pairwise losses, the result is even weaker: their zero values are even not sufficient for the zero value of (1-MAP).

To the best of our knowledge, there was no other theoretical result for the pairwise/listwise losses. Given that the pairwise and listwise approaches are regarded as the state-of-the-art in learning to rank [3, 11], it is very meaningful and important to perform more comprehensive analysis on these two approaches.

## 3 Main results

In this section, we present our main results on the relationship between ranking measures and the pairwise/listwise losses. The basic conclusion is that many pairwise and listwise losses are upper bounds of a quantity which we call the essential loss, and the essential loss is an upper bound of both $(1-NDCG)$ and $(1-MAP)$. Furthermore, the zero value of the essential loss is a *sufficient* and *necessary* condition for the zero values of $(1-NDCG)$ and $(1-MAP)$.

### 3.1 Essential loss: ranking as a sequence of classifications

In this subsection, we describe the *essential loss* for ranking.

First, we propose an alternative representation of the labels of objects (i.e., multi-level ratings). The basic idea is to construct a permutation set, with all the permutations in the set being *consistent* with the labels. The definition that a permutation is *consistent* with multi-level ratings is given as below.
**Definition 1.** *Given multi-level ratings $\mathcal{L}$ and permutation $y$, we say $y$ is **consistent** with $\mathcal{L}$, if $\forall i, s \in \{1, ..., n\}$ satisfying $i < s$, we always have $l(y(i)) \geq l(y(s))$, where $y(i)$ represents the index of the object that is ranked at the i-th position in $y$. We denote $Y_{\mathcal{L}} = \{y | y \text{ is consistent with } \mathcal{L}\}$.*

According to the definition, it is clear that the NDCG and MAP of a ranking function equal one, if and only if the ranked list (permutation) given by the ranking function is consistent with the labels.

Second, given each permutation $y \in Y_{\mathcal{L}}$, we decompose the ranking of objects $\mathbf{x}$ into several sequential steps. For each step $s$, we distinguish $x_{y(s)}$, the object ranked at the $s$-th position in $y$, from all the other objects ranked below the $s$-th position in $y$, using ranking function $f$.[5] Specifically, we denote $\mathbf{x}_{(s)} = \{x_{y(s)}, \cdots, x_{y(n)}\}$ and define a classifier based on $f$, whose target output is $y(s)$,

$$T_f(\mathbf{x}_{(s)}) = \arg \max_{j \in \{y(s), \cdots, y(n)\}} f(x_j). \tag{5}$$

It is clear that there are $n - s$ possible outputs of this classifier, i.e., $\{y(s), \cdots, y(n)\}$. The 0-1 loss for this classification task can be written as follows, where the second equality is based on the definition of $T_f$,

$$l_s\big(f; \mathbf{x}_{(s)}, y(s)\big) = I_{\{T_f(\mathbf{x}_{(s)}) \neq y(s)\}} = 1 - \prod_{i=s+1}^{n} I_{\{f(x_{y(s)}) > f(x_{y(i)})\}}.$$

We give a simple example in Figure 1 to illustrate the aforementioned process of decomposition.

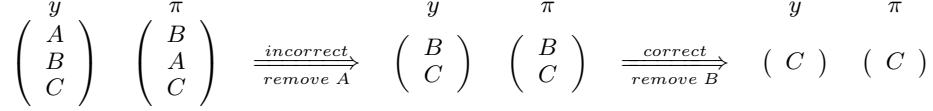

Figure 1: Modeling ranking as a sequence of classifications

Suppose there are three objects, $A$, $B$, and $C$, and a permutation $y = (A, B, C)$. Suppose the output of the ranking function for these objects is $(2, 3, 1)$, and accordingly the predicted ranked list is $\pi = (B, A, C)$. At step one of the decomposition, the ranking function predicts object $B$ to be on the top of the list. However, $A$ should be on the top according to $y$. Therefore, a prediction error occurs. For step two, we remove $A$ from both $y$ and $\pi$. Then the ranking function predicts object $B$ to be on the top of the remaining list. This is in accordance with $y$ and there is no prediction error. After that, we further remove object $B$, and it is easy to verify there is no prediction error in step three either. Overall, the ranking function makes one error in this sequence of classification tasks.

Third, we assign a non-negative weight $\beta(s)(s = 1, \cdots, n - 1)$ to the classification task at the $s$-th step, representing its importance to the entire sequence. We compute the weighted sum of the classification errors of all individual tasks,

$$L_\beta(f; \mathbf{x}, y) \triangleq \sum_{s=1}^{n-1} \beta(s)\big(1 - \prod_{i=s+1}^{n} I_{\{f(x_{y(s)}) > f(x_{y(i)})\}}\big), \tag{6}$$

and then define the minimum value of the weighted sum over all the permutations in $Y_\mathcal{L}$ as the *essential loss for ranking*.

$$L_\beta(f; \mathbf{x}, \mathcal{L}) = \min_{y \in Y_\mathcal{L}} L_\beta(f; \mathbf{x}, y). \tag{7}$$

According to the above definition of the essential loss, we can obtain its following nice property. Denote the ranked list produced by $f$ as $\pi_f$. Then it is easy to verify that,

$$L_\beta(f; \mathbf{x}, \mathcal{L}) = 0 \iff \exists y \in Y_\mathcal{L} \text{ satisfying } L_\beta(f; \mathbf{x}, y) = 0 \iff \pi_f = y \in Y_\mathcal{L}.$$

In other words, the essential loss is zero if and only if the permutation given by the ranking function is consistent with the labels. Further considering the discussions on the consistent permutation at the begining of this subsection, we can come to the conclusion that the zero value of the essential loss is a sufficient and necessary condition for the zero values of (1-NDCG) and (1-MAP).

## 3.2 Essential loss: upper bound of measure-based ranking errors

In this subsection, we show that the essential loss is an upper bound of $(1 - \text{NDCG})$ and $(1 - \text{MAP})$, when specific weights $\beta(s)$ are used.

**Theorem 1.** *Given $K$-level rating data $(\mathbf{x}, \mathcal{L})$ with $n_k$ objects having label $k$ and $\sum_{i=k^*}^{K} n_i > 0$, then $\forall f$, the following inequalities hold,*

(1)   $1 - NDCG(f; \mathbf{x}, \mathcal{L}) \leq \dfrac{1}{N_n} L_{\beta_1}(f; \mathbf{x}, \mathcal{L})$, where $\beta_1(s) = G\big(l(y(s))\big)D(s), \forall y \in Y_\mathcal{L}$;

(2)   $1 - MAP(f; \mathbf{x}, \mathcal{L}) \leq \dfrac{1}{\sum_{i=k^*}^{K} n_i} L_{\beta_2}(f; \mathbf{x}, \mathcal{L})$, where $\beta_2(s) \equiv 1$.

*Proof.* (1) We now prove the inequality for $(1 - \text{NDCG})$. First, we reformulate NDCG using the permutation set $Y_\mathcal{L}$. This can be done by changing the index of the sum in NDCG from the rank

position $r$ in $\pi_f$ to the rank position $s$ in $\forall y \in Y_{\mathcal{L}}$. Considering that $s = y^{-1}(\pi_f(r))$ and $r = \pi_f^{-1}(y(s))$, it is easy to verify,

$$NDCG(f;\mathbf{x},\mathcal{L}) = \frac{1}{N_n} \sum_{s=1}^{n} G\Big(l\big(\pi_f(\pi_f^{-1}y(s))\big)\Big) D\big(\pi_f^{-1}(y(s))\big) = \frac{1}{N_n} \sum_{s=1}^{n} G\big(l(y(s))\big) D\big(\pi_f^{-1}(y(s))\big).$$

Second, we consider the essential loss case by case. Note that

$$L_{\beta_1}(f;\mathbf{x},\mathcal{L}) = \min_{y \in Y_{\mathcal{L}}} \sum_{s=1}^{n-1} G\big(l(y(s))\big) D(s) \Big(1 - \prod_{i=s+1}^{n} I_{\{\pi_f^{-1}(y(s)) < \pi_f^{-1}(y(i))\}}\Big).$$

Then $\forall y \in Y_{\mathcal{L}}$, if position $s$ satisfies $\prod_{i=s+1}^{n} I_{\{\pi_f^{-1}(y(s)) < \pi_f^{-1}(y(i))\}} = 1$ (i.e., $\forall i > s$, $\pi_f^{-1}(y(s)) < \pi_f^{-1}(y(i))$), we have $\pi_f^{-1}(y(s)) \leq s$. As a consequence, $D(s) \prod_{i=s+1}^{n} I_{\{\pi_f^{-1}(y(s)) < \pi_f^{-1}(y(i))\}} = D(s) \leq D\big(\pi_f^{-1}(y(s))\big)$. Otherwise, if $\prod_{i=s+1}^{n} I_{\{\pi_f^{-1}(y(s)) < \pi_f^{-1}(y(i))\}} = 0$, it is easy to see that $D(s) \prod_{i=s+1}^{n} I_{\{\pi_f^{-1}(y(s)) < \pi_f^{-1}(y(i))\}} = 0 \leq D\big(\pi_f^{-1}(y(s))\big)$. To sum up, $\forall s \in \{1,2,...,n-1\}$, $D(s) \prod_{i=s+1}^{n} I_{\{\pi_f^{-1}(y(s)) < \pi_f^{-1}(y(i))\}} \leq D\big(\pi_f^{-1}(y(s))\big)$. Further considering $\pi_f^{-1}(y(n)) \leq n$ and $D(\cdot)$ is a decreasing function, we have $D(n) \leq D\big(\pi_f^{-1}(y(n))\big)$. As a result, we obtain,

$$1 - NDCG(f;\mathbf{x},\mathcal{L}) = \frac{1}{N_n} \sum_{s=1}^{n} G\big(l(y(s))\big) \Big(D(s) - D\big(\pi_f^{-1}(y(s))\big)\Big) \leq \frac{1}{N_n} L_{\beta_1}(f;\mathbf{x},\mathcal{L}).$$

(2) We then prove the inequality for $(1-\text{MAP})$. First, we prove the result for 2-level ratings. Given 2-level rating data $(\mathbf{x},\mathcal{L})$, it can be proved (see Lemma 1 in [4]) that $L_{\beta_2}(f;\mathbf{x},\mathcal{L}) = n_1 - i_0 + 1$, where $i_0$ denotes the position of the first object with label 0 in $\pi_f$, and $i_0 \leq n_1 + 1$. We then consider $n_1\big(1 - MAP(f;\mathbf{x},\mathcal{L})\big) = n_1 - \sum_{s:\ l(\pi_f(s))=1} \frac{\sum_{i \leq s} I_{\{l(\pi_f(i))=1\}}}{s}$ case by case. If $i_0 > n_1$ (i.e., the first object with label 0 is ranked after position $n_1$ in $\pi_f$), then all the objects with label 1 are ranked before the objects with label 0. Thus $n_1(1 - MAP(f;\mathbf{x},\mathcal{L})) = n_1 - n_1 = 0 = L_{\beta_2}(f;\mathbf{x},\mathcal{L})$. If $i_0(\pi_f) \leq n_1$, there are $i_0(\pi_f) - 1$ objects with label 1 ranked before all the objects with label 0. Thus $n_1(1 - MAP(f;\mathbf{x},\mathcal{L})) \leq n_1 - i_0(\pi_f) + 1 = L_{\beta_2}(f;\mathbf{x},\mathcal{L})$. This proves the theorem for 2-level ratings.

Second, given $K$-level rating data $(\mathbf{x},\mathcal{L})$, we denote the 2-level ratings induced by $\mathcal{L}$ as $\mathcal{L}'$. Then it is easy to verify $Y_{\mathcal{L}} \subseteq Y_{\mathcal{L}'}$. As a result, we have,

$$L_{\beta_2}(f;\mathbf{x},\mathcal{L}') = \min_{y \in Y_{\mathcal{L}'}} L_{\beta_2}(f;\mathbf{x},y) \leq \min_{y \in Y_{\mathcal{L}}} L_{\beta_2}(f;\mathbf{x},y) = L_{\beta_2}(f;\mathbf{x},\mathcal{L}).$$

Using the result for 2-level ratings, we obtain

$$1 - MAP(f;\mathbf{x},\mathcal{L}) = 1 - MAP(f;\mathbf{x},\mathcal{L}') \leq \frac{1}{\sum_{i=k^*}^{K-1} n_i} L_{\beta_2}(f;\mathbf{x},\mathcal{L}') \leq \frac{1}{\sum_{i=k^*}^{K-1} n_i} L_{\beta_2}(f;\mathbf{x},\mathcal{L}).$$

$\square$

### 3.3 Essential loss: lower bound of loss functions

In this section, we show that many pairwise/listwise losses are upper bounds of the essential loss.

**Theorem 2.** *The pairwise losses in Ranking SVM, RankBoost, and RankNet, and the listwise loss in ListMLE are all upper bounds of the essential loss, i.e.,*

$$(1) \quad L_\beta(f;\mathbf{x},\mathcal{L}) \leq \big(\max_{1 \leq s \leq n-1} \beta(s)\big) L^p(f;\mathbf{x},\mathcal{L});$$

$$(2) \quad L_\beta(f;\mathbf{x},\mathcal{L}) \leq \frac{1}{\ln 2}\big(\max_{1 \leq s \leq n-1} \beta(s)\big) L^l(f;\mathbf{x},y), \forall y \in Y_{\mathcal{L}}.$$

*Proof.* (1) We now prove the inequality for the pairwise losses. First, we reformulate the pairwise losses using permutation set $Y_{\mathcal{L}}$,

$$L^p(f;\mathbf{x},\mathcal{L}) = \sum_{s=1}^{n-1} \sum_{\substack{i=s+1, \\ l(y(s)) \neq l(y(i))}}^{n} \phi\big(f(x_{y(s)}) - f(x_{y(i)})\big) = \sum_{s=1}^{n-1} \sum_{i=s+1}^{n} a\big(y(i),y(s)\big) \phi\big(f(x_{y(s)}) - f(x_{y(i)})\big),$$

where $y$ is an arbitrary permutation in $Y_{\mathcal{L}}$, $a(i,j) = 1$ if $l(i) \neq l(j)$; $a(i,j) = 0$ otherwise. Note that only those pairs whose first object has a larger label than the second one are counted in the pairwise loss. Thus, the value of the pairwise loss is equal $\forall y \in Y_{\mathcal{L}}$.

Second, we consider the value of $a\big(T_f(\mathbf{x}_{(s)}), y(s)\big)$ case by case. $\forall y$ and $\forall s \in \{1, 2, ..., n-1\}$, if $a\big(T_f(\mathbf{x}_{(s)}), y(s)\big) = 1$ (i.e., $\exists i_0 > s$, satisfying $l(y(i_0)) \neq l(y(s))$ and $f(x_{y(i_0)}) > f(x_{y(s)})$), considering that function $\phi$ in Ranking SVM, RankBoost and RankNet are all non-negative, non-increasing, and $\phi(0) = 1$, we have,

$$\sum_{i=s+1}^{n} a\big(y(i), y(s)\big)\phi\big(f(x_{y(s)}) - f(x_{y(i)})\big)$$

$$\geq a\big(y(i_0), y(s)\big)\phi\big(f(x_{y(s)}) - f(x_{y(i_0)})\big) = \phi\big(f(x_{y(s)}) - f(x_{y(i_0)})\big) > 1 = a\big(T_f(\mathbf{x}_{(s)}), y(s)\big).$$

If $a\big(T_f(\mathbf{x}_{(s)}), y(s)\big) = 0$, it is clear that $\sum_{i=s+1}^{n} a\big(y(i), y(s)\big)\phi\big(f(x_{y(s)}) - f(x_{y(i)})\big) \geq 0 = a\big(T_f(\mathbf{x}_{(s)}), y(s)\big)$. Therefore,

$$\sum_{s=1}^{n-1} \beta(s) \sum_{i=s+1}^{n} a\big(y(i), y(s)\big)\phi\big(f(x_{y(s)}) - f(x_{y(i)})\big) \geq \sum_{s=1}^{n-1} \beta(s)a\big(T_f(\mathbf{x}_{(s)}), y(s)\big). \tag{8}$$

Third, it can be proved (see Lemma 2 in [4]) that the following inequality holds,

$$L_\beta(f; \mathbf{x}, \mathcal{L}) \leq \max_{y \in Y_{\mathcal{L}}} \sum_{s=1}^{n-1} \beta(s)a\big(T_f(\mathbf{x}_{(s)}), y(s)\big).$$

Considering inequality (8) and noticing that the pairwise losses are equal $\forall y \in Y_{\mathcal{L}}$, we have

$$L_\beta(f; \mathbf{x}, \mathcal{L}) \leq \max_{y \in Y_{\mathcal{L}}} \sum_{s=1}^{n-1} \beta(s) \sum_{i=s+1}^{n} a\big(y(i), y(s)\big)\phi\big(f(x_{y(s)}) - f(x_{y(i)})\big) \leq \big(\max_{1 \leq s \leq n-1} \beta(s)\big) L^p(f; \mathbf{x}, \mathcal{L}).$$

(2) We then prove the inequality for the loss function of ListMLE. Again, we prove the result case by case. Consider the loss of ListMLE in Eq.(3). $\forall y$ and $\forall s \in \{1, 2, ..., n-1\}$, if $I_{\{T_f(\mathbf{x}_{(s)}) \neq y(s)\}} = 1$ (i.e., $\exists i_0 > s$ satisfying $f(x_{y(i_0)}) > f(x_{y(s)})$), then $e^{f(x_{y(s)})} < \frac{1}{2}\sum_{i=s}^{n} e^{f(x_{y(s)})}$. Therefore, we have $-\ln \frac{e^{f(x_{y(s)})}}{\sum_{i=s}^{n} e^{f(x_{y(i)})}} > \ln 2 = \ln 2\, I_{\{T_f(\mathbf{x}_{(s)}) \neq y(s)\}}$. If $I_{\{T_f(\mathbf{x}_{(s)}) \neq y(s)\}} = 0$, then it is clear $-\ln \frac{e^{f(x_{y(s)})}}{\sum_{i=s}^{n} e^{f(x_{y(i)})}} > 0 = \ln 2\, I_{\{T_f(\mathbf{x}_{(s)}) \neq y(s)\}}$. To sum up, we have,

$$\sum_{s=1}^{n-1} \beta(s)\Big(-\ln \frac{e^{f(x_{y(s)})}}{\sum_{i=s}^{n} e^{f(x_{y(i)})}}\Big) > \sum_{s=1}^{n-1} \beta(s) \ln 2\, I_{\{T_f(\mathbf{x}_{(s)}) \neq y(s)\}} \geq \ln 2 \min_{y \in Y_{\mathcal{L}}} L_\beta(\pi_f, y) = \ln 2\, L_\beta(\pi_f, \mathcal{L}).$$

By further relaxing the inequality, we obtain the following result,

$$L_\beta(f; \mathbf{x}, \mathcal{L}) \leq \frac{1}{\ln 2}\big(\max_{1 \leq s \leq n-1} \beta(s)\big) L^l(f; \mathbf{x}, y), \forall y \in Y_{\mathcal{L}}.$$

$\square$

### 3.4 Summary

We have the following inequalities by combining the results obtained in the previous subsections.

(1) The pairwise losses in Ranking SVM, RankBoost, and RankNet are upper bounds of $(1-\text{NDCG})$ and $(1-\text{MAP})$.

$$1 - NDCG(f; \mathbf{x}, \mathcal{L}) \leq \frac{G(K-1)D(1)}{N_n} L^p(f; \mathbf{x}, \mathcal{L});$$

$$1 - MAP(f; \mathbf{x}, \mathcal{L}) \leq \frac{1}{\sum_{i=k^*}^{K} n_i} L^p(f; \mathbf{x}, \mathcal{L}).$$

(2) The listwise loss in ListMLE is an upper bound of $(1-\text{NDCG})$ and $(1-\text{MAP})$.

$$1 - NDCG(f; \mathbf{x}, \mathcal{L}) \leq \frac{G(K-1)D(1)}{N_n \ln 2} L^l(f; \mathbf{x}, y), \forall y \in Y_{\mathcal{L}};$$

$$1 - MAP(f; \mathbf{x}, \mathcal{L}) \leq \frac{1}{\ln 2 \sum_{i=k^*}^{K} n_i} L^l(f; \mathbf{x}, y), \forall y \in Y_{\mathcal{L}}.$$

Table 1: Ranking accuracy on OHSUMED

| Methods | RankNet | W-RankNet | ListMLE | W-ListMLE |
|---|---|---|---|---|
| NDCG@5 | 0.4568 | **0.4868** | 0.4471 | **0.4588** |
| NDCG@10 | 0.4414 | **0.4604** | 0.4347 | **0.4453** |

| Methods | Regression | Ranking SVM | RankBoost | FRank | ListNet | SVMMAP |
|---|---|---|---|---|---|---|
| NDCG@5 | 0.4278 | 0.4164 | 0.4494 | 0.4588 | 0.4432 | 0.4516 |
| NDCG@10 | 0.4110 | 0.414 | 0.4302 | 0.4433 | 0.441 | 0.4319 |

## 4  Discussion

The proofs of Theorems 1 and 2 actually suggest a way to improve existing loss functions. The key idea is to introduce weights related to $\beta_1(s)$ to the loss functions so as to make them tighter bounds of $(1-\text{NDCG})$.

Specifically, we introduce weights to the pairwise and listwise losses in the following way,

$$\tilde{L}^p(f; \mathbf{x}, \mathcal{L}) = \sum_{s=1}^{n-1} G(l(y(s))) D\Big(1 + \sum_{k=l(y(s))+1}^{K-1} n_k\Big) \sum_{i=s+1}^{n} a\big(y(i), y(s)\big) \phi\big(f(x_{y(s)}) - f(x_{y(i)})\big), \forall y \in Y_{\mathcal{L}};$$

$$\tilde{L}^l(f; \mathbf{x}, y) = \sum_{s=1}^{n-1} G(l(y(s))) D(s) \Big(-f(x_{y(s)}) + \ln\big(\sum_{i=s}^{n} \exp(f(x_{y(i)}))\big)\Big).$$

It can be proved (see Proposition 1 in [4]) that the above weighted losses are still upper bounds of $(1-\text{NDCG})$ and they are lower bounds of the original pairwise and listwise losses. In other words, the above weighted loss functions are tighter bounds of $(1-\text{NDCG})$ than existing loss functions.

We tested the effectiveness of the weighted loss functions on the OHSUMED dataset in LETOR 3.0.[6] We took RankNet and ListMLE as example algorithms. The methods that minimize the weighted loss functions are referred to as W-RankNet and W-ListMLE. From Table 1, we can see that (1) W-RankNet and W-ListMLE significantly outperform RankNet and ListMLE. (2) W-RankNet and W-ListMLE also outperform other baselines on LETOR such as Regression, Ranking SVM, Rank-Boost, FRank [15], ListNet and SVMMAP [18]. These experimental results seem to indicate that optimizing tighter bounds of the ranking measures can lead to better ranking performances.

## 5  Conclusion and future work

In this work, we have proved that many pairwise/listwise losses in learning to rank are actually upper bounds of measure-based ranking errors. We have also shown a way to improve existing methods by introducing appropriate weights to their loss functions. Experimental results have validated our theoretical analysis. As future work, we plan to investigate the following issues.

(1) We have modeled ranking as a sequence of classifications, when defining the essential loss. We believe this modeling has its general implication for ranking, and will explore its other usages.

(2) We have taken NDCG and MAP as two examples in this work. We will study whether the essential loss is an upper bound of other measure-based ranking errors.

(3) We have taken the loss functions in Ranking SVM, RankBoost, RankNet and ListMLE as examples in this study. We plan to investigate the loss functions in other pairwise and listwise ranking methods, such as RankCosine [13], ListNet [3], FRank [15] and QBRank [19].

(4) While we have mainly discussed the upper-bound relationship in this work, we will study whether loss functions in existing learning-to-rank methods are statistically consistent with the essential loss and the measure-based ranking errors.

## Footnotes

[2]Note that recently people try to directly optimize ranking measures [17, 12, 14, 18]. The relationship between ranking measures and the loss functions in such work is explicitly known. However, for other methods, the relationship is unclear.

[3]For example, for information retrieval, $\mathbf{x}$ represents the documents associated with a query.

[4]Note that the bounds given in the original papers of [5] and [10] are with respect to DCG. Here we give their equivalent forms in terms of NDCG, and set $P(\cdot|x_i, S) = \delta_{l(i)}(\cdot)$ in the bound of [5], for ease of comparison.

[5]For simplicity and clarity, we assume $f(x_i) \neq f(x_j) \; \forall i \neq j$, such that the classifier will have a unique output. It can be proved (see [4]) that the main results in this paper still hold without this assumption.

[6] http://research.microsoft.com/~letor

# References

[1] R. Baeza-Yates and B. Ribeiro-Neto. *Modern Information Retrieval*. Addison Wesley, May 1999.

[2] C. Burges, T. Shaked, E. Renshaw, A. Lazier, M. Deeds, N. Hamilton, and G. Hullender. Learning to rank using gradient descent. In *ICML '05: Proceedings of the 22nd International Conference on Machine learning*, pages 89–96, New York, NY, USA, 2005. ACM.

[3] Z. Cao, T. Qin, T.-Y. Liu, M.-F. Tsai, and H. Li. Learning to rank: from pairwise approach to listwise approach. In *ICML '07: Proceedings of the 24th International Conference on Machine learning*, pages 129–136, New York, NY, USA, 2007. ACM.

[4] W. Chen, T.-Y. Liu, Y. Lan, Z. Ma, and H. Li. Essential loss: Bridge the gap between ranking measures and loss functions in learning to rank. Technical report, Microsoft Research, MSR-TR-2009-141, 2009.

[5] D. Cossock and T. Zhang. Statistical analysis of bayes optimal subset ranking. *Information Theory*, 54:5140–5154, 2008.

[6] Y. Freund, R. Iyer, R. E. Schapire, and Y. Singer. An efficient boosting algorithm for combining preferences. *Journal of Machine Learning Research*, 4:933–969, 2003.

[7] R. Herbrich, K. Obermayer, and T. Graepel. Large margin rank boundaries for ordinal regression. In *Advances in Large Margin Classifiers*, pages 115–132, Cambridge, MA, 1999. MIT.

[8] K. Järvelin and J. Kekäläinen. Cumulated gain-based evaluation of ir techniques. *ACM Transactions on Information Systems*, 20(4):422–446, 2002.

[9] T. Joachims. Optimizing search engines using clickthrough data. In *KDD '02: Proceedings of the 8th ACM SIGKDD international conference on Knowledge discovery and data mining*, pages 133–142, New York, NY, USA, 2002. ACM.

[10] P. Li, C. Burges, and Q. Wu. Mcrank: Learning to rank using multiple classification and gradient boosting. In *NIPS '07: Advances in Neural Information Processing Systems 20*, pages 897–904, Cambridge, MA, 2008. MIT.

[11] T.-Y. Liu, J. Xu, T. Qin, W.-Y. Xiong, and H. Li. Letor: Benchmark dataset for research on learning to rank for information retrieval. In *SIGIR '07 Workshop*, San Francisco, 2007. Morgan Kaufmann.

[12] Q. L. Olivier Chapelle and A. Smola. Large margin optimization of ranking measures. In *NIPS workshop on Machine Learning for Web Search 2007*, 2007.

[13] T. Qin, X.-D. Zhang, M.-F. Tsai, D.-S. Wang, T.-Y. Liu, , and H. Li. Query-level loss functions for information retrieval. *Information Processing and Management*, 44(2):838–855, 2008.

[14] M. Taylor, J. Guiver, S. Robertson, and T. Minka. Softrank: optimizing non-smooth rank metrics. In *Proceedings of the International Conference on Web search and web data mining*, pages 77–86, Palo Alto, California, USA, 2008. ACM.

[15] M.-F. Tsai, T.-Y. Liu, T. Qin, H.-H. Chen, and W.-Y. Ma. Frank: a ranking method with fidelity loss. In *SIGIR '07: Proceedings of the 30th annual ACM SIGIR conference*, pages 383–390, Amsterdam, The Netherlands, 2007. ACM.

[16] F. Xia, T.-Y. Liu, J. Wang, W. Zhang, and H. Li. Listwise approach to learning to rank - theory and algorithm. In *ICML '08: Proceedings of the 25th International Conference on Machine learning*, pages 1192–1199. Omnipress, 2008.

[17] J. Xu and H. Li. Adarank: a boosting algorithm for information retrieval. In *SIGIR '07: Proceedings of the 30th annual international ACM SIGIR conference on Research and development in information retrieval*, pages 391–398, 2007.

[18] Y. Yue, T. Finley, F. Radlinski, and T. Joachims. A support vector method for optimizing average precision. In *SIGIR '07: Proceedings of the 30th annual international ACM SIGIR conference on Research and development in information retrieval*, pages 271–278, New York, NY, USA, 2007. ACM.

[19] Z. Zheng, H. Zha, T. Zhang, O. Chapelle, K. Chen, and G. Sun. A general boosting method and its application to learning ranking functions for web search. In *NIPS '07: Advances in Neural Information Processing Systems 20*, pages 1697–1704. MIT, Cambridge, MA, 2008.

